# Bayesian Probabilistic Co-Subspace Addition

**Lei Shi**
Baidu.com, Inc
`shilei06@baidu.com`

## Abstract

For modeling data matrices, this paper introduces Probabilistic Co-Subspace Addition (PCSA) model by simultaneously capturing the dependent structures among both rows and columns. Briefly, PCSA assumes that each entry of a matrix is generated by the additive combination of the linear mappings of two low-dimensional features, which distribute in the row-wise and column-wise latent subspaces respectively. In consequence, PCSA captures the dependencies among entries intricately, and is able to handle non-Gaussian and heteroscedastic densities. By formulating the posterior updating into the task of solving Sylvester equations, we propose an efficient variational inference algorithm. Furthermore, PCSA is extended to tackling and filling missing values, to adapting model sparseness, and to modelling tensor data. In comparison with several state-of-art methods, experiments demonstrate the effectiveness and efficiency of Bayesian (sparse) PCSA on modeling matrix (tensor) data and filling missing values.

## 1 Introduction

This paper focuses on modeling data matrices by simultaneously capturing the dependent structures among both rows and columns, which is especially useful for filling missing values. Using Gaussian Process (GP), Xu *et al* [25] modified the kernel to incorporate relational information and drew outputs from GPs. Widely used in geostatistics, Linear Models of Corregionalization (LMC) [5] learns the covariance structures over the vectorized data matrix. In [12, 16], Bayesian probabilistic matrix factorization (PMF) is investigated via modeling the row-wise and column-wise specific variances and inferred based on suitable priors. Probabilistic Matrix Addition (PMA) [1] describes the covariance structures among rows and columns, showing promising results compared with GP regression, PMF and LMC. However, both LMC and PMA are inefficient on large scale matrices.

On high dimensional data, subspace structures are usually designed in statistical models with reduced numbers of free parameters, leading to improvement on both learning efficiency and accuracy [3, 11, 24]. Equipping PMA with the subspace structures, this paper proposes a simple yet novel generative Probabilistic Co-Subspace Addition (PCSA) model, which, as its name, assumes that all entries in a matrix come from the sums of linear mappings of latent features in row-wise and column-wise hidden subspaces. Including many existing models as its special cases (see Section 2.1), PCSA is able to capture the dependencies among entries intricately, fit the non-Gaussian and heteroscedastic density, and extract the hidden features in the co-subspaces.

We propose a variational Bayesian algorithm for inferring both the parameters and the latent dimensionalities of PCSA. For quick and stable convergence, we formulate the posterior updating procedure into solving Sylvester equations [10]. Furthermore, Bayesian PCSA is implemented in three extensions. First, missing values in data matrices are easily tackled and filled by iterating with the variational inference. Second, with a Jeffreys prior, Bayesian sparse PCSA is implemented with an adaptive model sparseness [4]. Finally, we extend the PCSA on matrix data (i.e., 2nd-order tensor) to PCSA-$k$ for modelling tensor data with an arbitrary order $k$.

On the task of filling missing values in matrix data, we compare (sparse) PCSA with several state-of-art models/approaches, including PMA, Robust Bayesian PMF and Bayesian GPLVM [21]. The datasets under consideration range from multi-label classification data, user-item rating data for collaborative filtering, and face images. Further on tensor structured face image data, PCSA is compared with the $\mathrm{M}^2\mathrm{SA}$ method [6] that uses consecutive SVDs on all modes of the tensor. Although simple and not designed for any particular application, through experiments PCSA shows results promisingly comparable to or better than the competing approaches.

## 2 PCSA Model and Variational Bayesian Inference

### 2.1 Probabilistic Co-Subspace Addition (PCSA)

The PCSA model defines distributions over real valued matrices. Letting $\mathbf{X} \in \mathcal{R}^{D_1 \times D_2}$ be an observed matrix with $D_1 \leq D_2$ without loss of generality[1], we start by outlining a generative model for $\mathbf{X}$. Consider two hidden variables $\mathbf{y} \sim \mathcal{N}(\mathbf{y}|\mathbf{0}_{d_1}, \mathbf{I}_{d_1})$ and $\mathbf{z} \sim \mathcal{N}(\mathbf{z}|\mathbf{0}_{d_2}, \mathbf{I}_{d_2})$ with $d_1 < D_1$ and $d_2 < D_2$, where $\mathbf{0}_d$ denotes a $d$-dim vector with all entries being zeros and $\mathbf{I}_d$ denotes a $d \times d$ identity matrix. Using the concatenation nomenclature of Matlab, two matrices of hidden factors $\mathbf{Y} = [\mathbf{y}_{*1}, \ldots, \mathbf{y}_{*D_2}] \in \mathcal{R}^{d_1 \times D_2}$ and $\mathbf{Z} = [\mathbf{z}_{*1}, \ldots, \mathbf{z}_{*D_1}] \in \mathcal{R}^{d_2 \times D_1}$ are column-wise independently generated, respectively. Through two linear mapping matrices $\mathbf{A} \in \mathcal{R}^{D_1 \times d_1}$ and $\mathbf{B} \in \mathcal{R}^{D_2 \times d_2}$, each entry $x_{ij} \in \mathbf{X}$ is independent given $\mathbf{Y}$ and $\mathbf{Z}$ by $x_{ij} = \mathbf{a}_{i*}\mathbf{y}_{*j} + \mathbf{b}_{j*}\mathbf{z}_{*i} + e_{ij}$, where $\mathbf{a}_{i*}$ is the $i$-th row of $\mathbf{A}$. Each $e_{ij} \sim \mathcal{N}(e_{ij}|0, 1/\tau)$ is independently Gaussian distributed and independent from $\mathbf{Y}, \mathbf{Z}$. The generative process of $\mathbf{X}$ thus is:

- Get $\mathbf{Y}$ by independently drawing each vector $\mathbf{y}_{*j} \sim \mathcal{N}(\mathbf{y}_{*j}|\mathbf{0}_{d_1}, \mathbf{I}_{d_1})$ for $j = 1, \ldots, D_2$;
- Get $\mathbf{Z}$ by independently drawing each vector $\mathbf{z}_{*i} \sim \mathcal{N}(\mathbf{z}_{*i}|\mathbf{0}_{d_2}, \mathbf{I}_{d_2})$ for $i = 1, \ldots, D_1$;
- Get $\mathbf{E} \in \mathcal{R}^{D_1 \times D_2}$ by independently drawing each element $e_{ij} \sim \mathcal{N}(e_{ij}|0, 1/\tau)$ for $\forall i, j$;
- Get $\mathbf{X} = \mathbf{AY} + (\mathbf{BZ})^\top + \mathbf{E}$ given $\mathbf{Y}$ and $\mathbf{Z}$, i.e., additively combines the co-subspaces.

Given parameters $\boldsymbol{\theta} = \{\mathbf{A}, \mathbf{B}, \tau\}$, the joint distribution of $\mathbf{X}$, $\mathbf{Y}$ and $\mathbf{Z}$ is $p(\mathbf{X}, \mathbf{Y}, \mathbf{Z}|\boldsymbol{\theta}) =$

$$[\prod_{j=1}^{D_2} \mathcal{N}(\mathbf{y}_{*j}|\mathbf{0}_{d_1}, \mathbf{I}_{d_1})] \cdot [\prod_{i=1}^{D_1} \mathcal{N}(\mathbf{z}_{*i}|\mathbf{0}_{d_2}, \mathbf{I}_{d_2})] \cdot [\prod_{i=1}^{D_1} \prod_{j=1}^{D_2} \mathcal{N}(x_{ij}|\mathbf{a}_{i*}\mathbf{y}_{*j} + \mathbf{b}_{j*}\mathbf{z}_{*i}, 1/\tau)]. \quad (1)$$

**Properties and relations to existing work.** Albeit its simple generative process, PCSA owns meaningful properties and can be viewed as an extension of several existing models.

- **Intricate dependencies between entries in $\mathbf{X}$.** Although each entry $x_{ij} \in \mathbf{X}$ is independent given $\mathbf{Y}$ and $\mathbf{Z}$, the PCSA model captures the dependencies along rows as well as columns in the joint $\mathbf{X}$. Particularly, assuming $D_1$ is the data dimensionality while $D_2$ is the sample size, the samples (column vectors) in $\mathbf{X}$ is dependent from each other by PCSA. When $\mathbf{B}$ is constrained as $\mathbf{0}$, PCSA will degenerate to Probabilistic PCA (PPCA) [3], which insists the sample i.i.d. assumption.

- **Non-Gaussianity and heteroscedasticity.** If we still consider $D_1$ as the data dimensionality and $D_2$ as the sample size, the PCSA model handles the non-Gaussianity in samples of $\mathbf{X}$. As an extreme example, if all columns of $\mathbf{Z}^\top \mathbf{B}^\top$ are discretized to take values from a set of $n$ vectors, PCSA degenerates to Mixture of PPCA [7, 20] with $n$ components, whose subspace loadings are the same. That is, learning such a PCSA model actually implements the group PPCA [24] throughout different components. Also, if marginalizing $\mathbf{Z}$, we are describing the column samples of $\mathbf{X}$ with a dependent heteroscedasticity.

- **Co-subspace feature extraction.** Although able to describe the row-wise and column-wise covariances, PMA [1] requires estimating and inverting two (large) kernel matrices with sizes $D_1 \times D_1$ and $D_2 \times D_2$ respectively, which is intractable for many real world applications. In contrast, PCSA has $(D_1 d_1 + D_2 d_2 + 1)$ free parameters and inverts smaller matrices, and recovers PMA when $d_1 = D_1$ and $d_2 = D_2$. Moreover, PCSA is able to extract the hidden features $\mathbf{Y}$ and $\mathbf{Z}$ simultaneously.

## 2.2 Variational Bayesian Inference

Given $\mathbf{X}$ and the hidden dimensionalities $(d_1, d_2)$, we can estimate PCSA's parameters $\boldsymbol{\theta} = \{\mathbf{A}, \mathbf{B}, \tau\}$ by maximizing the likelihood $p(\mathbf{X}|\boldsymbol{\theta})$. However, the capacity control is essential to generalization ability, for which we proceed to deliver a variational Bayesian inference on PCSA.

By introducing hyper-parameters $\boldsymbol{\varsigma} = [\varsigma_1, \ldots, \varsigma_{d_1}]^\top$ and $\boldsymbol{\varphi} = [\varphi_1, \ldots, \varphi_{d_2}]^\top$ for a *hierarchical Normal-Gamma prior* on $(\mathbf{A}, \boldsymbol{\varsigma})$ and $(\mathbf{B}, \boldsymbol{\varphi})$ respectively [3, 7, 20], we have the prior $p(\boldsymbol{\theta})$ as

$$p(\boldsymbol{\theta}, \boldsymbol{\varsigma}, \boldsymbol{\varphi}) = p(\tau)p(\mathbf{A}, \boldsymbol{\varsigma})p(\mathbf{B}, \boldsymbol{\varphi}), \qquad p(\tau) = \Gamma(\tau|u^\tau, v^\tau),$$

$$p(\mathbf{A}, \boldsymbol{\varsigma}) = p(\mathbf{A}|\boldsymbol{\varsigma})p(\boldsymbol{\varsigma}), \quad p(\mathbf{A}|\boldsymbol{\varsigma}) = \prod_{i=1}^{d_1} \mathcal{N}(\mathbf{a}_{*i}|\mathbf{0}_{D_1}, \mathbf{I}_{D_1}/\varsigma_i), \; p(\boldsymbol{\varsigma}) = \prod_{i=1}^{d_1} \Gamma(\varsigma_i|u_i^\varsigma, v_i^\varsigma),$$

$$p(\mathbf{B}, \boldsymbol{\varphi}) = p(\mathbf{B}|\boldsymbol{\varphi})p(\boldsymbol{\varphi}), \; p(\mathbf{B}|\boldsymbol{\varphi}) = \prod_{i=1}^{d_2} \mathcal{N}(\mathbf{b}_{*i}|\mathbf{0}_{D_2}, \mathbf{I}_{D_2}/\varphi_i), \; p(\boldsymbol{\varphi}) = \prod_{i=1}^{d_2} \Gamma(\varphi_i|u_i^\varphi, v_i^\varphi), \text{ (2)}$$

where $\Gamma(\cdot|u, v)$ denotes a Gamma distribution with a shape parameter $u$ and an inverse scale parameter $v$. Each column $\mathbf{a}_{*i}$ of the mapping matrix $\mathbf{A}$ *priori* independently follows a spherical Gaussian with a precision scalar $\varsigma_i$, i.e., an automatic relevance determination (ARD) type prior [14]. Each precision $\varsigma_i$ further follows a Gamma prior for completing the specification of the Bayesian model.

It is computationally intractable to evaluate the marginal likelihood $p(\mathbf{X}) = \int p(\mathbf{X}|\boldsymbol{\Theta})p(\boldsymbol{\Theta})\mathrm{d}\boldsymbol{\Theta}$, where $\boldsymbol{\Theta} = \{\mathbf{Z}, \mathbf{Y}, \boldsymbol{\theta}, \boldsymbol{\varsigma}, \boldsymbol{\varphi}\}$ represents the set of all parameters and latent variables. Since MCMC samplers are inefficient for high dimensional data, this paper chooses variational inference instead [11], which introduces a distribution $Q(\boldsymbol{\Theta})$ and approximates maximizing the log marginal likelihood $\log p(\mathbf{X})$ by maximizing a lower bound $\mathcal{L}(Q) = \int Q(\boldsymbol{\Theta}) \log \frac{p(\mathbf{X}, \boldsymbol{\Theta})}{Q(\boldsymbol{\Theta})} \mathrm{d}\boldsymbol{\Theta}$. For tractability, $Q(\boldsymbol{\Theta})$ is factorized into the following mean-field form:

$$Q(\boldsymbol{\Theta}) = Q(\mathbf{Y})Q(\mathbf{Z})Q(\mathbf{A})Q(\mathbf{B})Q(\tau)Q(\boldsymbol{\varsigma})Q(\boldsymbol{\varphi}), \;\; Q(\mathbf{Y}) = \prod_{i=1}^{D_2} Q(\mathbf{y}_{*i}), \;\; Q(\mathbf{Z}) = \prod_{i=1}^{D_1} Q(\mathbf{z}_{*i}),$$

$$Q(\mathbf{A}) = \prod_{i=1}^{d_1} Q(\mathbf{a}_{*i}), \quad Q(\mathbf{B}) = \prod_{i=1}^{d_2} Q(\mathbf{b}_{*i}), \quad Q(\boldsymbol{\varsigma}) = \prod_{i=1}^{d_1} Q(\varsigma_i), \quad Q(\boldsymbol{\varphi}) = \prod_{i=1}^{d_2} Q(\varphi_i). \qquad \text{(3)}$$

Maximizing $\mathcal{L}(Q)$ w.r.t. the above $Q(\vartheta)$ for $\forall \vartheta \in \boldsymbol{\Theta}$ leads to the following explicit conjugate forms

$$Q(\mathbf{y}_{*t}) = \mathcal{N}(\mathbf{y}_{*t}|\bar{\mathbf{y}}_{*t}, \bar{\boldsymbol{\Sigma}}^Y), \; Q(\mathbf{z}_{*i}) = \mathcal{N}(\mathbf{z}_{*i}|\bar{\mathbf{z}}_{*i}, \bar{\boldsymbol{\Sigma}}^Z),$$
$$Q(\varsigma_i) = \Gamma(\varsigma_i|\bar{u}_i^\varsigma, \bar{v}_i^\varsigma), \; Q(\varphi_i) = \Gamma(\varphi_i|\bar{u}_i^\varphi, \bar{v}_i^\varphi),$$
$$Q(\mathbf{a}_{*i}) = \mathcal{N}(\mathbf{a}_{*i}|\bar{\mathbf{a}}_{*i}, \psi^A \mathbf{I}_{D_1}), \; Q(\mathbf{b}_{*i}) = \mathcal{N}(\mathbf{b}_{*i}|\bar{\mathbf{b}}_{*i}, \psi^B \mathbf{I}_{D_2}), \; Q(\tau) = \Gamma(\tau|\bar{u}^\tau, \bar{v}^\tau). \; \text{(4)}$$

For expression simplicity, we denote $\bar{\mathbf{A}} = [\bar{\mathbf{a}}_{*1}, \ldots, \bar{\mathbf{a}}_{*d_1}]$ and similarly for $\bar{\mathbf{B}}$, $\bar{\mathbf{Y}}$ and $\bar{\mathbf{Z}}$. During maximizing $\mathcal{L}(Q)$, the solutions of $\bar{\mathbf{Y}}$ and $\bar{\mathbf{Z}}$ are bundled and conditional on each other:

$$\bar{\mathbf{Y}} = \langle\tau\rangle\mathbf{S}^A\bar{\mathbf{A}}^\top(\mathbf{X} - \bar{\mathbf{Z}}^\top\bar{\mathbf{B}}^\top), \quad \bar{\boldsymbol{\Sigma}}^Y = \mathbf{S}^A, \quad \mathbf{S}^A = \left[(1 + \langle\tau\rangle D_1 \psi^A)\mathbf{I}_{d_1} + \langle\tau\rangle\bar{\mathbf{A}}^\top\bar{\mathbf{A}}\right]^{-1},$$

$$\bar{\mathbf{Z}} = \langle\tau\rangle\mathbf{S}^B\bar{\mathbf{B}}^\top(\mathbf{X} - \bar{\mathbf{A}}\bar{\mathbf{Y}})^\top, \quad \bar{\boldsymbol{\Sigma}}^Z = \mathbf{S}^B, \quad \mathbf{S}^B = \left[(1 + \langle\tau\rangle D_2 \psi^B)\mathbf{I}_{d_2} + \langle\tau\rangle\bar{\mathbf{B}}^\top\bar{\mathbf{B}}\right]^{-1}, \text{ (5)}$$

where $\langle\cdot\rangle$ is expectation and $\langle\tau\rangle = \bar{u}^\tau/\bar{v}^\tau$. Directly updating by the above converges neither quickly nor stably. Instead after putting one equation into the other, we attain a Sylvester equation [10] and can efficiently solve it by many tools. Then $\bar{\mathbf{Z}}$ is obtained by solving $\mathbf{L}_1^Z\bar{\mathbf{Z}}\mathbf{L}_2^Z - \bar{\mathbf{Z}} + \mathbf{L}_3^Z = \mathbf{0}$,

$$\text{with } \mathbf{L}_1^Z = \langle\tau\rangle^2\mathbf{S}^B\bar{\mathbf{B}}^\top\bar{\mathbf{B}}, \; \mathbf{L}_2^Z = \bar{\mathbf{A}}\mathbf{S}^A\bar{\mathbf{A}}^\top, \; \mathbf{L}_3^Z = \langle\tau\rangle\mathbf{S}^B\bar{\mathbf{B}}^\top\mathbf{X}^\top\left(\mathbf{I}_{D_1} - \langle\tau\rangle\bar{\mathbf{A}}\mathbf{S}^A\bar{\mathbf{A}}^\top\right), \quad \text{(6)}$$

whose solution is further put into Eq.(5) to update $\bar{\mathbf{Y}}^2$. Given $\bar{\mathbf{Y}}$ and $\bar{\mathbf{Z}}$, updating $(\bar{\mathbf{A}}, \bar{\mathbf{B}})$ is similar.

The remainders of $Q(\boldsymbol{\Theta})$ in Eq.(4) are updated as

$$\psi^A = d_1/\mathrm{tr}\left(\mathbf{S}^{Y^{-1}}\right), \qquad \mathbf{S}^Y = \left[\langle\tau\rangle\mathbf{K}^Y + \mathrm{diag}(\langle\boldsymbol{\varsigma}\rangle)\right]^{-1}, \qquad \mathbf{K}^Y = \bar{\mathbf{Y}}\bar{\mathbf{Y}}^\top + D_2\bar{\boldsymbol{\Sigma}}^Y,$$

$$\psi^B = d_2/\mathrm{tr}\left(\mathbf{S}^{Z^{-1}}\right), \qquad \mathbf{S}^Z = \left[\langle\tau\rangle\mathbf{K}^Z + \mathrm{diag}(\langle\boldsymbol{\varphi}\rangle)\right]^{-1}, \qquad \mathbf{K}^Z = \bar{\mathbf{Z}}\bar{\mathbf{Z}}^\top + D_1\bar{\boldsymbol{\Sigma}}^Z,$$

$$\bar{u}^\tau = u^\tau + \frac{D_1 D_2}{2}, \qquad \bar{\mathbf{u}}^\varsigma = \mathbf{u}^\varsigma + \frac{D_1}{2}\mathbf{1}_{d_1}, \qquad \bar{\mathbf{u}}^\varphi = \mathbf{u}^\varphi + \frac{D_2}{2}\mathbf{1}_{d_2},$$

$$\bar{v}^\tau = v^\tau + \frac{D_1}{2}\mathrm{tr}(\bar{\boldsymbol{\Sigma}}^Y \bar{\mathbf{A}}^\top \bar{\mathbf{A}} + \psi^A \mathbf{K}^Y) + \frac{D_2}{2}\mathrm{tr}(\bar{\boldsymbol{\Sigma}}^Z \bar{\mathbf{B}}^\top \bar{\mathbf{B}} + \psi^B \mathbf{K}^Z) + \frac{1}{2}||(\mathbf{X} - \bar{\mathbf{A}}\bar{\mathbf{Y}} - \bar{\mathbf{Z}}^\top \bar{\mathbf{B}}^\top)||_F^2,$$

$$\bar{\mathbf{v}}^\varsigma = \mathbf{v}^\varsigma + \frac{1}{2}\mathrm{diag}(\bar{\mathbf{A}}^\top \bar{\mathbf{A}}) + \frac{D_1 \psi^A}{2}\mathbf{1}_{d_1}, \qquad \bar{\mathbf{v}}^\varphi = \mathbf{v}^\varphi + \frac{1}{2}\mathrm{diag}(\bar{\mathbf{B}}^\top \bar{\mathbf{B}}) + \frac{D_2 \psi^B}{2}\mathbf{1}_{d_2}, \qquad (7)$$

where $\langle\varsigma\rangle = [\bar{u}_1^\varsigma/\bar{v}_1^\varsigma, \ldots, \bar{u}_{d_1}^\varsigma/\bar{v}_{d_1}^\varsigma]^\top$, $\langle\varphi\rangle = [\bar{u}_1^\varphi/\bar{v}_1^\varsigma, \ldots, \bar{u}_{d_2}^\varphi/\bar{v}_{d_2}^\varphi]^\top$, $\mathrm{tr}(\cdot)$ stands for trace, $\mathrm{diag}(\cdot)$ inter-converts between a vector and a diagonal matrix, and $||\cdot||_F$ is the Frobenius norm.

In implementation, all Gamma priors in Eq.(2) are set to be vague as $\Gamma(\cdot|10^{-3}, 10^{-3})$. During learning, redundant columns of $\bar{\mathbf{A}}$ and $\bar{\mathbf{B}}$ will be pushed to approach zeros, which actually makes Bayesian model selection on hidden dimensionalities $d_1$ and $d_2$.

## 3 Extensions

### 3.1 Filling Missing Values

In many real applications, $\mathbf{X}$ is usually partially observed with some missing entries. The goal here is to infer not only the PCSA model but also the missing values in $\mathbf{X}$ based on the model structure.

Similar to the settings of PMA in [1], let us begin with a full matrix $\widetilde{\mathbf{X}}$, where the missing values are randomly filled. We denote $\mathcal{M} = \{(i,j) : \tilde{x}_{ij} \text{ is missing}\}$ as the index set of the missing values therein. In each iteration, we "pretend" that $\widetilde{\mathbf{X}}$ is the observed matrix and update $Q(\boldsymbol{\Theta})$ by Eqs.(6~7). Then given $Q(\boldsymbol{\Theta})$, the missing entries $\{\tilde{x}_{ij} : (i,j) \in \mathcal{M}\}$ are updated by maximizing $\mathcal{L}(Q)$, i.e., $\tilde{x}_{ij} = \bar{x}_{ij}$ with $\bar{\mathbf{X}} = \arg\max_{\mathbf{X}} \mathcal{L}(Q) = \bar{\mathbf{A}}\bar{\mathbf{Y}} + \bar{\mathbf{Z}}^\top \bar{\mathbf{B}}^\top$. This updating manner plays a role of adaptive regularization [2], and performs well in experiments as to be shown in Section 4. Moreover, filling missing values in PMA [1] needs to infer the column and row factors by either Gibbs sampling or MAP. In contrast, PCSA directly employs $\bar{\mathbf{Y}} \cup \bar{\mathbf{Z}}$ that were estimated already in the variational inference, and thus saves the computing cost.

### 3.2 Bayesian Sparse PCSA

As discussed above, PCSA describes observations by mapping hidden features ($\mathbf{Y}$ and $\mathbf{Z}$) in the co-subspaces through $\mathbf{A}$ and $\mathbf{B}$ respectively, i.e., $\mathbf{A}$ and $\mathbf{B}$ serve similarly to the transformation matrix in Factor Analysis and PPCA. For high dimensional data, the parameters $\mathbf{A}$ and $\mathbf{B}$ probably suffer from inaccurate estimations and are difficult to interpret. Sparsification is one popularly-used method to improve model interpretability in the literature. In this part, we extend to provide a Bayesian treatment on the sparse PCSA model.

LASSO [19] encourages model sparseness by adding an $\ell_1$ regularizer, which is equivalent to a Laplacian prior. In [4], the sparseness is adaptively controlled by assigning a hierarchical Normal-Jeffreys (NJ) prior. Paper [9] showed that the NJ prior performs better than the Laplacian on sparse PPCA. In this paper, we choose to adopt the NJ prior for learning a sparse PCSA model.

Different from Eq.(2), each column of $\mathbf{A}$ and $\mathbf{B}$ follows a hierarchical Normal-Jeffreys prior:

$$p(\mathbf{A}|\boldsymbol{\alpha}^A) = \prod_{i=1}^{d_1} \mathcal{N}(\mathbf{a}_{*i}|\mathbf{0}, \alpha_i^A \mathbf{I}_{D_1}), \quad p(\boldsymbol{\alpha}^A) = \prod_{i=1}^{d_1} \frac{1}{\alpha_i^A}, \quad \text{with } \boldsymbol{\alpha}^A = [\alpha_1^A, \ldots, \alpha_{d_1}^A]^\top,$$

$$p(\mathbf{B}|\boldsymbol{\alpha}^B) = \prod_{i=1}^{d_2} \mathcal{N}(\mathbf{b}_{*i}|\mathbf{0}, \alpha_i^B \mathbf{I}_{D_2}), \quad p(\boldsymbol{\alpha}^B) = \prod_{i=1}^{d_2} \frac{1}{\alpha_i^B}, \quad \text{with } \boldsymbol{\alpha}^B = [\alpha_1^B, \ldots, \alpha_{d_2}^B]^\top, \quad (8)$$

which also encourages the variances in $\boldsymbol{\alpha}^A$ and $\boldsymbol{\alpha}^B$ of redundant dimensions to approach zeros.

The prior on $\tau$ remains the same as in Eq.(2). Still under the variational inference framework, we now let $\boldsymbol{\Theta} = \{\mathbf{Z}, \mathbf{Y}, \boldsymbol{\theta}\}$ and $Q(\boldsymbol{\Theta}) = Q(\mathbf{Y})Q(\mathbf{Z})Q(\boldsymbol{\theta})$ takes the conjugate form same as in Eq.(4). In consequence, we optimize $\mathcal{L}(Q; \boldsymbol{\alpha}^A, \boldsymbol{\alpha}^B) = \int Q(\boldsymbol{\Theta}) \log \frac{p(\mathbf{X}, \boldsymbol{\Theta}|\boldsymbol{\alpha}^A, \boldsymbol{\alpha}^B)}{Q(\boldsymbol{\Theta})} d\boldsymbol{\Theta}$ w.r.t. $Q(\boldsymbol{\Theta})$, $\boldsymbol{\alpha}^A$ and $\boldsymbol{\alpha}^B$, where $\mathcal{L}(Q; \boldsymbol{\alpha}^A, \boldsymbol{\alpha}^B) \leq \log p(\mathbf{X}|\boldsymbol{\alpha}^A, \boldsymbol{\alpha}^B)$. Posterior inference remains the same as above, except that all appearances of $\langle\varsigma\rangle$ and $\langle\varphi\rangle$ are replaced with $[1/\alpha_1^A, \ldots, 1/\alpha_{d_1}^A]^T$ and

$[1/\alpha_1^B, \ldots, 1/\alpha_{d_2}^B]^T$ respectively. Then given $Q(\boldsymbol{\Theta})$, the variances $\boldsymbol{\alpha}^A$ and $\boldsymbol{\alpha}^B$ are updated via $\boldsymbol{\alpha}^A = \frac{1}{D_1+2}[\mathrm{diag}(\bar{\mathbf{A}}^\top \bar{\mathbf{A}}) + \psi^A]$ and $\boldsymbol{\alpha}^B = \frac{1}{D_2+2}[\mathrm{diag}(\bar{\mathbf{B}}^\top \bar{\mathbf{B}}) + \psi^B]$.

### 3.3 Modeling High-Order Tensor Data

Up till now, we have been talking about modeling $\mathbf{X}$ when it is a matrix, and this part extends the PCSA model and its Bayesian inference to cover the cases when $\mathbf{X}$ is structured as a tensor. Tensors are higher-order generalizations of vectors (1st-order tensors) and matrices (2nd-order tensors) [6]. Each dimension of a tensor is called as a *mode*, and the *order* of a tensor is determined as the number of its modes. Let us denote tensors with open-face uppercase letters (e.g., $\mathbb{X}$, $\mathbb{Y}$, $\mathbb{Z}$), in comparison with the bold uppercase letters (e.g., $\mathbf{X}$, $\mathbf{Y}$, $\mathbf{Z}$) for matrices. A $k$th-order tensor $\mathbb{X}$ can be denoted by $\mathbb{X} \in \mathcal{R}^{D_1 \times D_2 \times \ldots \times D_k}$, where its dimensionalities in each mode are $D_1, D_2, \ldots, D_k$ respectively. An element and a (1st-mode) vector of $\mathbb{X}$ are denoted by $x_{j_1 j_2 \ldots j_k}$ and $\mathbf{x}_{*j_2 \ldots j_k}$ respectively, where $1 \le j_i \le D_i$ for each $i = 1, \ldots, k$. Moreover, the 1st-mode *flattening* transform of $\mathbb{X}$, denoted by $\mathbf{F}(\mathbb{X}) \in \mathcal{R}^{D_1 \times (D_2 D_3 \ldots D_k)}$, is obtained by concatenating all the (1st-mode) vectors of $\mathbb{X}$. Vice versa, a $([D_1, \ldots, D_k])$-*tensorization* of a matrix $\mathbf{X} \in \mathcal{R}^{D_1 \times (D_2 \ldots D_k)}$ is defined as $\mathbb{T}(\mathbf{X}, [D_1, \ldots, D_k]) \in \mathcal{R}^{D_1 \times D_2 \times \ldots \times D_k}$, so that $\mathbb{T}(\mathbf{F}(\mathbb{X}), [D_1, \ldots, D_k]) = \mathbb{X}$. An $i$th *mode-shift* transform is defined as $\mathbb{M}(\mathbb{X}, i) \in \mathcal{R}^{D_i \times D_{i+1} \times \ldots \times D_k \times D_1 \times \ldots \times D_{i-1}}$, which shifts the modes sequentially in a cycle and until the $i$th-mode in $\mathbb{X}$ becomes the 1st-mode in $\mathbb{M}(\mathbb{X}, i)$.

Based on the above definitions, the PCSA model describes a $k$th-order tensor data $\mathbb{X}^{D_1 \times \ldots \times D_k}$ through the following generative process: (i) for each mode $i$, all elements of the hidden tensor $\mathbb{Y}^{(i)} \in \mathcal{R}^{d_i \times D_{i+1} \times \ldots \times D_k \times D_1 \times \ldots \times D_{i-1}}$ are assumed i.i.d. drawn from $\mathcal{N}(y_{j_i j_{i+1} \ldots j_k j_1 j_{i-1}}^{(i)} | 0, 1)$; (ii) draw each element $x_{j_1 \ldots j_k} \sim \mathcal{N}(x_{j_1 \ldots j_k} | \sum_{i=1}^{k} \mathbf{a}_{j_i *}^{(i)} \mathbf{y}_{*j_{i+1} \ldots j_k j_1 \ldots j_{i-1}}^{(i)}, 1/\tau)$, i.e., $\mathbb{X}$ is actually generated by a mode-shift co-subspace addition:

$$\mathbb{X} = \mathbb{E} + \sum_{i=1}^{k} \mathbb{M}\left(\mathbb{T}\left(\bar{\mathbf{X}}^{(i)}, [D_i, \ldots, D_k, D_1, \ldots, D_{i-1}]\right), k+2-i\right), \tag{9}$$

where each $\bar{\mathbf{X}}^{(i)} = \mathbf{A}^{(i)} \mathbf{F}(\mathbb{Y}^{(i)})$ and the matrix $\mathbf{A}^{(i)} \in \mathcal{R}^{D_i \times d_i}$ maps $\mathbb{Y}^{(i)}$ to $\mathbb{X}$. Shortly named as PCSA-$k$, this model has latent tensors $\{\mathbb{Y}^{(i)}\}_{i=1}^{k}$ and parameters $\boldsymbol{\theta} = \{\tau\} \cup \{\mathbf{A}^{(i)}\}_{i=1}^{k}$ with latent scales $\{d_i\}_{i=1}^{k}$. When $k = 2$, PCSA-2 is exactly the PCSA in Section 2.1 on matrix data. Also, it can be imagined as a kind of group Factor Analysis [24].

Same as Eq.(2), each column of $\mathbf{A}^{(i)}$ takes a hierarchical Normal-Gamma prior, and the Bayesian inference in Section 2.2 can be trivially extended for covering PCSA-$k$ model. Please see the details in the supplementary materials. Except the involvement of the tensor structure and its operators, there is another difference compared with the variational posterior updating based on a matrix $\mathbf{X}$. Remembering $(Q(\mathbf{Y}), Q(\mathbf{Z}))$ and $(Q(\mathbf{A}), Q(\mathbf{B}))$ pairwise were decoupled and updated by solving Sylvester equations, we can decouple neither $\{Q(\mathbb{Y}^{(i)})\}_i$ nor $\{Q(\mathbf{A}^{(i)})\}_i$ into Sylvester equations for the general $k > 2$. Instead, sequentially for each $i = 1, \ldots, k$, we update only $Q(\mathbb{Y}^{(i)})$ (or $Q(\mathbf{A}^{(i)})$) and keep the remaining $\{Q(\mathbb{Y}^{(u)})\}_{u \ne i}$ (or $\{Q(\mathbf{A}^{(u)})\}_{u \ne i}$) fixed.

## 4 Experimental Results

### 4.1 Predicting Missing Entries in Weight Matrices

**On Emotions and CAL500 Data.** The proposed PCSA model can be viewed as a rather direct extension of the PMA model, which showed advantages over GPR, LMC and PMF in [1]. Following [1], the first experiment compares PCSA with PMA in filling the missing entries of a truncated log-odds matrix in multi-label classification. For $n$ samples and $m$ classes, the class memberships can be represented as an $n \times m$ binary matrix $\mathbf{G}$. A truncated log-odds matrix $\mathbf{X}$ is constructed with $x_{ij} = c$ if $g_{ij} = 1$ and $x_{ij} = -c$ if $g_{ij} = 0$, where $c$ is nonzero constant. In experiments, certain entries $x_{ij}$ are assumed to be missing and filled as $\tilde{x}_{ij}$ by an algorithm, and the performance is evaluated by the class membership prediction accuracy based on $\mathrm{sign}(\tilde{x}_{ij})$.

Two multi-label classification datasets are under consideration, namely Emotions [22] and CAL500 [23]. Already used in [1], the Emotions contains 593 samples with 72 numeric at-

tributes in 6 classes, and the number of classes that each sample belongs to ranges from 1 to 3. The constructed $\mathbf{X}$ for Emotions is thus $593 \times 6$. The CAL500 contains 502 samples with 68 numeric attributes in 174 classes, and the min and max numbers of classes that each sample belongs to are 13 and 48 respectively. The constructed $\mathbf{X}$ for CAL500 is thus $502 \times 174$, i.e., its size is larger and more balanced than the one for Emotions.

To test the capability in dealing with missing values, the proportion of the missing labels is increased from $10\%$ to $50\%$, with $5\%$ as a step size. Instead of Gibbs sampling, the MAP inference is used in PMA implementation for a fair comparison. After 10 independent runs on each dataset, Fig.1 reports the error rates for recovering the missing labels in the truncated log-odds matrices, by Bayesian PCSA, Bayesian sparse PCSA and PMA. On the relatively unbalanced Emotions data, PCSA outperforms sparse PCSA when the missing proportion is no larger than $40\%$, while sparse PCSA takes over the advantage when too many entries are missing due to the increasing importance of model sparsity. On the more balanced CAL500 data, the sparse PCSA keeps a slight outperformance over PCSA, again due to the sparsity. Moreover, PCSA and sparse PCSA always perform considerably better than PMA on both datasets. Table 1 reports the average time cost, where sparse PCSA shows a little quicker convergence than PCSA. Both are much quicker than PMA, since they do not need to either invert large covariances or infer the factor during filling missing values (see Section 3.1).

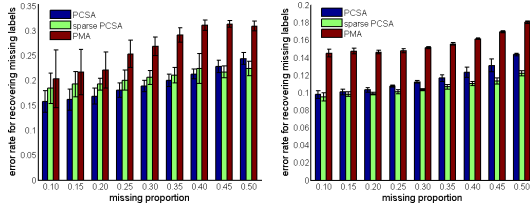

| dataset: | Emotions | CAL500 |
|---|---|---|
| PCSA | 4.0 | 17.3 |
| sparse PCSA | 3.5 | 11.6 |
| PMA | 22.9 | 198.3 |

Figure 1: Error rates of 10 independent runs for recovering the missing labels in Emotions (left) and CAL500 (right) data.

Table 1: Average time cost (in seconds) on each dataset throughout 10 independent runs and all missing proportions.

**On MovieLens and JesterJoke Data.** In many real applications, e.g. collaborative filtering, the size of the matrix $\mathbf{X}$ is much larger than the above. We proceed to consider on two larger weight datasets: the MovieLens100K data[3] and the JesterJoke3 data [8]. Particularly, the MovieLens100K dataset contains 100K ratings of 943 users on 1682 movies, which are ordinal values on the scale $[1, 5]$. The JesterJoke3 data contains ratings of 24983 users who have rated between 15 and 35 pieces of the total 100 jokes, where the ratings are continuous in $[-10.0, 10.0]$.

Recently in [12], Robust Bayesian Matrix Factorization (RBMF) was proposed by adopting a Student-$t$ prior in probabilistic matrix factorization, and showed promising results on predicting entries on both MovieLens100K and JesterJoke3 data. Following [12], in each run we randomly choose $70\%$ of the ratings for training, and use the remaining ratings as the missing values for testing. Given the true test ratings $\{r_t\}_{t=1}^T$ and the predictions $\{\tilde{r}_t\}_{t=1}^T$, the performance is evaluated based on the *rooted mean squared error* (RMSE), i.e., $\text{RMSE} = \sqrt{\frac{1}{T} \sum_{t=1}^T (r_t - \tilde{r}_t)^2}$, and the *mean absolute error* (MAE), i.e., $\text{MAE} = \frac{1}{T} \sum_{t=1}^T |r_t - \tilde{r}_t|$.

After 10 independent runs, the average RMSE and MAE values obtained by (sparse) PCSA are reported in Table 2, in comparison with the best results by RBMF (i.e., RBMF-RR) collected from [12]. Since PMA runs inefficiently on high dimensional data as in Table 1, it is not considered to fill the ratings in this experiment. It is observed that the performance by PCSA on predicting ratings is comparable with RBMF. On both RMSE and MAE scores, the sparse PCSA further improves the correctness and performs similarly to or better than RBMF.

Table 2: Average RMSE and MAE on MovieLens100K (left) and JesterJoke3 (right).

| model | RMSE | MAE | model | RMSE | MAE |
|---|---|---|---|---|---|
| PCSA | 0.903 | 0.708 | PCSA | 4.446 | 3.447 |
| sparse PCSA | **0.898** | 0.706 | sparse PCSA | **4.413** | **3.434** |
| RBMF-RR [12] | 0.900 | **0.705** | RBMF-RR [12] | 4.454 | 3.439 |

## 4.2 Completing Partially Observed Images

We consider two greyscale face image datasets, namely Frey [15] and ORL [17]. Specifically, Frey has 1965 images of size $28 \times 20$ taken from one person, and the data $\mathbf{X}$ is thus a $560 \times 1965$ matrix; ORL has 400 images of size $64 \times 64$ taken from 40 persons (10 images per person), and the data $\mathbf{X}$ is thus a $4096 \times 400$ matrix. Applied on these matrices, the PCSA model is expected to extract the latent correlations among pixels and images. In [13], Neil Lawrence proposed a Gaussian process latent variable model (GPLVM) for modeling and visualizing high dimensional data. Recently a Bayesian GPLVM [21] was developed and showed much improved performance on filling pixels in partially observed Frey faces. This experiment compares PCSA with Bayesian GPLVM[4].

While PCSA can utilize the partial observed samples, the Bayesian GPLVM cannot. Thus in each run, we randomly pick $n_f$ images as fully observed, and a half pixels of the remaining images are further randomly chosen as missing values. Same as [21], Bayesian GPLVM uses the $n_f$ images for training and then infers the missing pixels. In contrast, (sparse) PCSA uses all images as a whole matrix. In order to test the robustness, the $n_f$ for Frey is decreased gradually from 1000 to 200, and for ORL is decreased gradually from 300 to 50. Performance is evaluated by the *correlation coefficient* (CORR) and the MAE between the filled pixels and the ground truth.

On Frey and ORL data respectively, Fig.2 and Fig.3 report the CORR and MAE values of 10 independent runs by PCSA, sparse PCSA and Bayesian GPLVM. Both PCSA and sparse PCSA perform more accurately than Bayesian GPLVM in completing the missing pixels, and PCSA gives the best matching. Also, (sparse) PCSA shows promising stability against the decreasing fully observed sample size $n_f$, and this tendency is kept even when we assign *all* images are partially observed (i.e., $n_f = 0$), as exemplified by Fig.4. The results by Bayesian GPLVM deteriorate more obviously, because the partially observed images have no contribution during learning. Furthermore, the advantage of PCSA becomes more significant, as we shift from the Frey data for a single person, to the ORL data for multiple persons. It indirectly reflects the importance of extracting the correlations among different images, rather than keeping them independent. Sparse PCSA performs worse than PCSA in this task, mainly because it leads to a little too many sparse dimensions.

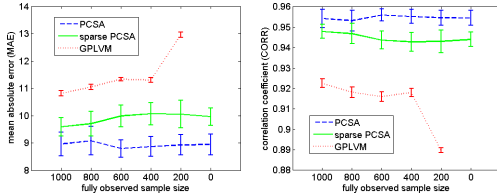

Figure 2: Results of 10 runs on Frey faces.

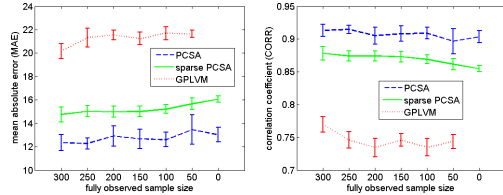

Figure 3: Results of 10 runs on ORL faces.

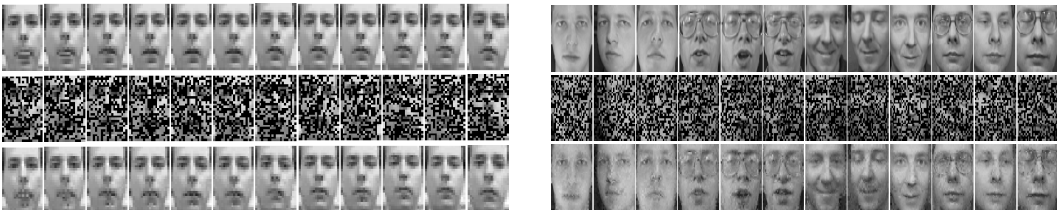

Figure 4: Reconstruction examples by PCSA when all images are partially observed: Frey (left) and ORL (right). Three rows from top are true, observed, and reconstructed images, respectively.

## 4.3 Completing Partially Observed Image Tensor

We proceed to consider modeling the face image data arranged in a tensor. The dataset under consideration is a subset of the CMU PIE database [18], and totally has 5100 face images from 30 individuals. Each person's face exhibits 170 images corresponding to 170 different pose-and-illumination combinations. Each normalized image has $32 \times 32$ greyscale pixels, and the dataset is thus a tensor $\mathbb{X} \in \mathcal{R}^{1024 \times 30 \times 170}$, whose three modes correspond to pixel, identity, and pose/illumination, respectively. Figure 5 shows some image examples of two persons. The PCSA-$k$ model (with $k = 3$ on the

3rd-order tensor $\mathbb{X}$) in Section 3.3 is expected to extract the co-subspace structures (i.e., correlations among pixels, identities, and poses/illuminations respectively) and fill the missing values in $\mathbb{X}$. In [6], an $\mathrm{M}^2\mathrm{SA}$ method was proposed to conduct multilinear subspace analysis with missing values on the tensor data, via consecutive SVD dimension reductions on each mode.

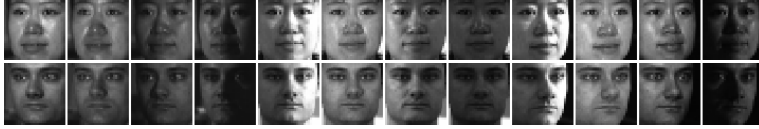

Figure 5: Typical normalized face images from the CMU `PIE` database.

true:

filled:

true:

filled:

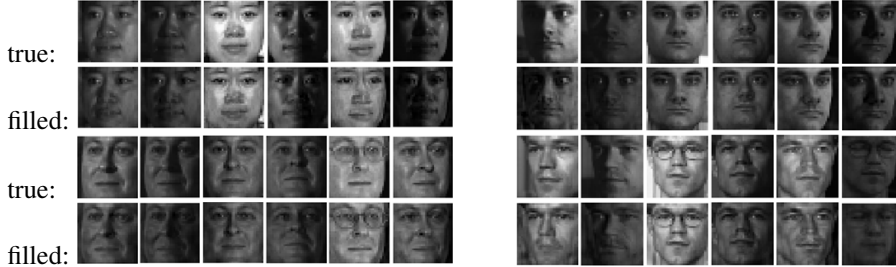

Figure 6: Typical missing images filled by PCSA-3. Original images (in the odd rows) are randomly picked and removed, and PCSA-3 fills the images in the even rows.

Table 3: Average CORR (left) and MAE (right) of 10 runs by PCSA-3 and $\mathrm{M}^2\mathrm{SA}$ on the CMU `PIE` data.

| missing proportion: | 10% | 20% | 30% |
|---|---|---|---|
| PCSA-3 | 0.937 | 0.926 | 0.908 |
| $\mathrm{M}^2\mathrm{SA}$ | 0.928 | 0.914 | 0.893 |
| missing proportion: | 10% | 20% | 30% |
| PCSA-3 | 14.6 | 18.3 | 21.5 |
| $\mathrm{M}^2\mathrm{SA}$ | 17.8 | 21.9 | 24.8 |

Here, the randomly drawn missing values are not pixels as in Section 4.2 but *images*. Compared with the true missing images, the goodness of the filled missing images is evaluated again by CORR and MAE. Still to test the capability in dealing with missing values, the proportion of the missing images is considered as 10%, 20% and 30%, respectively. After 10 independent runs for each proportion, the averages CORR and average MAE of filing the missing images by PCSA-3 and $\mathrm{M}^2\mathrm{SA}$ are compared in Table 3. During implementing $\mathrm{M}^2\mathrm{SA}$, the ratio of the subspace rank over the original rank is set as 0.6 according to Fig.9 in [6]. As shown in Table 3, PCSA-3 achieves the better performance in all cases. For demonstration, Fig.6 shows some filled missing images when the missing proportion is 20%, which match the original images steadily well.

## 5 Concluding Remarks

We have introduced the Probabilistic Co-Subspace Addition (PCSA) model, which simultaneously captures the dependent structures among both rows and columns in data matrices (tensors). Variational inference is proposed on PCSA for an approximate Bayesian learning, and the posteriors can be efficiently and stably updated by solving Sylvester equations. Capable to fill missing values, PCSA is extended to not only sparse PCSA with the help of a Jeffreys prior, but also PCSA-$k$ that models arbitrary $k$th-order tensor data. Although somewhat simple and not designed for any particular application, the experiments demonstrate the effectiveness and efficiency of PCSA on modeling matrix (tensor) data and filling missing values. The performance by PCSA may be further improved by considering nonlinear mappings with the kernel trick, which however is not that direct due to the coupling inner products between the co-subspaces.

**Acknowledgments**

The author would like to thank the anonymous reviewers for their useful comments on this paper.

## Footnotes

[1]Otherwise, we can transpose $\mathbf{X}$. This assumption is for efficient Sylvester equation solving in the sequel.

[2] The choice of computing $\bar{\mathbf{Z}}$ first is based on the assumption $D_1 \leq D_2$ for learning efficiency.

[3]Downloaded from www.grouplens.org/node/73.

[4]We use the code in http://staffwww.dcs.shef.ac.uk/people/N.Lawrence/vargplvm/.

# References

[1] A. Agovic, A. Banerjee, and S. Chatterjee. Probabilistic matrix addition. In *Proc. ICML*, pages 1025–1032, 2011.

[2] C. M. Bishop. Training with noise is equivalent to Tikhonov regularization. *Neural Computation*, 7(1):108–116, 1995.

[3] C. M. Bishop. Variational principal components. In *Proc. ICANN'1999*, volume 1, pages 509–514, 1999.

[4] M. A. T. Figueiredo. Adaptive sparseness using Jeffreys prior. In *Advances in NIPS*, volume 14, pages 679–704. MIT Press, Cambridge, MA, 2002.

[5] A. E. Gelfand and S. Banerjee. Multivariate spatial process models. In A. E. Gelfand, P. Diggle, P.Guttorp, and M. Fuentes, editors, *Handbook of Spatial Statistics*. CRC Press, 2010.

[6] X. Geng, K. Smith-Miles, Z.-H. Zhou, and L. Wang. Face image modeling by multilinear subspace analysis with missing values. *IEEE Trans. Syst., Man, Cybern. B, Cybern.*, 41(3):881–892, 2011.

[7] Z. Ghahramani and G. Hinton. The EM algorithm for mixtures of factor analyzers. Technical Report CRG-TR-96-1, Department of Computer Science, University of Toronto, Toronto, Canada, 1997.

[8] K. Goldberg, T. Roeder, D. Gupta, and C. Perkins. Eigentaset: A constant time collaborative filtering algorithm. *Information Retrieval*, 4(2):133–151, 2001.

[9] Y. Guan and J. Dy. Sparse probabilistic principal component analysis. In *Proc. AISTATS'2009, JMLR W&CP*, volume 5, pages 185–192. 2009.

[10] D. Y. Hu and L. Reichel. Krylov-subspace methods for the Sylvester equation. *Linear Algebra and Its Applications*, 172:283–313, 1992.

[11] M. I. Jordan, editor. *Learning in graphical models*. MIT Press, Cambridge MA, 1999.

[12] B. Lakshimanarayan, G. Bouchard, and C. Archambeau. Robust Bayesian matrix factorisation. In *Proc. AISTATS'2011, JMLR W&CP*, volume 15, pages 425–433. 2011.

[13] N. D. Lawrence. Gaussian process latent variable models for visualisation of high dimensional data. In *Advances in NIPS*, volume 16, pages 329–336. MIT Press, Cambridge, MA, 2003.

[14] R. M. Neal. *Bayesian Learning for Neural Networks*. Springer-Verlag, New York, 1996.

[15] S. Roweis, L. K. Saul, and G. Hinton. Global coordination of local linear models. In *Advances in NIPS*, volume 14, pages 889–896. MIT Press, Cambridge, MA, 2002.

[16] R. Salakhutdinov and A. Mnih. Probabilistic matrix factorization. In *Advances in NIPS*, volume 20, pages 1257–1264. MIT Press, Cambridge, MA, 2008.

[17] F. Samaria and A. Harter. Parameterisation of a stochastic model for human face identification. In *Proc. 2nd IEEE Workshop on Applications of Computer Vision*, pages 138–142, 1994.

[18] T. Sim, S. Baker, and M. Bsat. The CMU pose, illumination, and expression database. *IEEE Trans. Patten Anal. Mach. Intell.*, 25(12):1615–1618, 2003.

[19] R. Tibshirani. Regression shrinkage and selection via the lasso. *J. R. Stat. Soc. B*, 58(1):267–288, 1996.

[20] M. E. Tipping and C. M. Bishop. Mixtures of probabilistic principal component analyzers. *Neural Computation*, 11(2):443–482, 1999.

[21] M. Titsias and N. Lawrence. Bayesian Gaussian process latent variable model. In *Proc. AISTATS'2009, JMLR W&CP*, volume 9, pages 844–851. 2010.

[22] K. Trohidis, G. Tsoumakas, G. Kalliris, and I. Vlahavas. Multilabel classification of music into emotions. In *Proc. Intl. Conf. on Music Information Retrieval (ISMIR)*, pages 325–330, 2008.

[23] D. Turnbull, L. Barrington, D. Torres, and G. Lanckriet. Semantic annotation and retrieval of music and sound effects. *IEEE Trans. Audio, Speech and Lang. Process.*, 16(2):467–476, 2008.

[24] S. Virtanen, A. Klami, S. A. Khan, and S. Kaski. Bayesian group factor analysis. In *Proc. AISTATS'2012, JMLR W&CP*, volume 22, pages 1269–1277. 2012.

[25] Z. Xu, K. Kersting, and V. Tresp. Multi-relational learning with Gaussian processes. In *Proc. IJCAI'2009*, pages 1309–1314, 2009.

